# Heterogeneous Multitask Learning with Joint Sparsity Constraints

**Xiaolin Yang**
Department of Statistics
Carnegie Mellon University
Pittsburgh, PA 15213
xyang@stat.cmu.edu

**Seyoung Kim**
Machine Learning Department
Carnegie Mellon University
Pittsburgh, PA 15213
sssykim@cs.cmu.edu

**Eric P. Xing**
Machine Learning Department
Carnegie Mellon University
Pittsburgh, PA 15213
epxing@cs.cmu.edu

## Abstract

Multitask learning addresses the problem of learning related tasks that presumably share some commonalities on their input-output mapping functions. Previous approaches to multitask learning usually deal with homogeneous tasks, such as purely regression tasks, or entirely classification tasks. In this paper, we consider the problem of learning multiple related tasks of predicting both continuous and discrete outputs from a common set of input variables that lie in a high-dimensional feature space. All of the tasks are related in the sense that they share the same set of relevant input variables, but the amount of influence of each input on different outputs may vary. We formulate this problem as a combination of linear regressions and logistic regressions, and model the joint sparsity as $L_1/L_\infty$ or $L_1/L_2$ norm of the model parameters. Among several possible applications, our approach addresses an important open problem in genetic association mapping, where the goal is to discover genetic markers that influence multiple correlated traits jointly. In our experiments, we demonstrate our method in this setting, using simulated and clinical asthma datasets, and we show that our method can effectively recover the relevant inputs with respect to all of the tasks.

## 1 Introduction

In multitask learning, one is interested in learning a set of related models for predicting multiple (possibly) related outputs (i.e., tasks) given a set of input variables [4]. In many applications, the multiple tasks share a common input space, but have different functional mappings to different output variables corresponding to different tasks. When the tasks and their corresponding models are believed to be related, it is desirable to learn all of the models jointly rather than treating each task as independent of each other and fitting each model separately. Such a learning strategy that allows us to borrow information across tasks can potentially increase the predictive power of the learned models.

Depending on the type of information shared among the tasks, a number of different algorithms have been proposed. For example, hierarchical Bayesian models have been applied when the parameter values themselves are thought to be similar across tasks [2, 14]. A probabilistic method for modeling the latent structure shared across multiple tasks has been proposed [16]. For problems of which the input lies in a high-dimensional space and the goal is to recover the shared sparsity structure across tasks, a regularized regression method has been proposed [10].

In this paper, we consider an interesting and not uncommon scenario of multitask learning, where the tasks are *heterogeneous* and bear a *union support*. That is, each task can be either a regression or classification problem, with the inputs lying in a very high-dimensional feature space, but only a small number of the input variables (i.e., predictors) are relevant to each of the output variables (i.e.,

responses). Furthermore, we assume that all of the related tasks possibly share common relevant predictors, but with varying amount of influence on each task.

Previous approaches for multitask learning usually consider a set of homogeneous tasks, such as regressions only, or classifications only. When each of these discrete or continuous prediction tasks is treated separately, given a high-dimensional design, the *lasso* method that penalizes the loss function with an $L_1$ norm of the parameters has been a popular approach for variable selection [13, 11], since the $L_1$ regularization has the property of shrinking parameters corresponding to irrelevant predictors exactly to zero. One of the successful extensions of the standard lasso is the group lasso that uses an $L_1/L_2$ penalty defined over predictor groups [15], instead of just the $L_1$ penalty ubiquitously over all predictors. Recently, a more general $L_1/L_q$-regularized regression scheme with $q > 0$ has been thoroughly investigated [17]. When the $L_1/L_q$ penalty is used in estimating the regression function for a single predictive task, it makes use of information about the grouping of input variables, and applies the $L_1$ penalty over the $L_q$ norm of the regression coefficients for each group of inputs. As a result, variable selection can be effectively achieved on each group rather than on each individual input variable. This type of regularization scheme can be also used against the output variables in a single classification task with multi-way (rather than binary) prediction, where the output is expanded from univariate to multivariate with dummy variables for each prediction category. In this situation the group lasso can promote selecting the same set of relevant predictors across all of the dummy variables (which is desirable since these dummy variables indeed correspond to only a single multi-way output). In our multitask learning problem, when the $L_1/L_2$ penalty of group lasso is used for multitask regression [9, 10, 1], the $L_2$ norm is applied to the regression coefficients for each input across all tasks, and the $L_1$ norm is applied to these $L_2$ norms, playing the role of selecting common input variables relevant to one or more tasks via a *sparse union support* recovery. Since the parameter estimation problem formulated with such penalty terms has a convex objective function, many of the algorithms developed for a general convex optimization problem can be used for solving the learning problem. For example, an interior point method and a preconditioned conjugate gradient algorithm have been used to solve a large-scale $L_1$-regularized linear regression and logistic regression [8]. In [6, 13], a coordinate-descent method was used in solving an $L_1$-regularized linear regression and generalized linear models, where the soft thresholding operator gives a closed-form solution for each coordinate in each iteration.

In this paper, we consider the more challenging, but realistic scenario of having heterogenous outputs, i.e., both continuous and discrete responses, in multitask learning. This means that the tasks in question consist of both regression and classification problems. Assuming a linear regression for continuous-valued output and a logistic regression for discrete-valued output with dummy variables for multiple categories, an $L_1/L_q$ penalty can be used to learn both types of tasks jointly for a sparse union support recovery. Since the $L_1/L_q$ penalty selects the same relevant inputs for all dummy outputs for each classification task, the desired consistency in chosen relevant inputs across the dummy variables corresponding to the same multi-way response is automatically maintained. We consider particular cases of $L_1/L_q$ regularizations with $q = 2$ and $q = \infty$.

Our work is primarily motivated by the problem of genetic association mapping based on genome-wide genotype data of single nucleotide polymorphisms (SNPs), and phenotype data such as disease status, clinical traits, and microarray data collected over a large number of individuals. The goal in this study is to identify the SNPs (or inputs) that explain the variation in the phenotypes (or outputs), while reducing false positives in the presence of a large number of irrelevant SNPs from the genome-scale data. Since many clinical traits for a given disease are highly correlated, it is greatly beneficial to combine information across multiple such related phenotypes because the inputs often involve millions of SNPs and the association signals of causal (or relevant) SNPs tend to be very weak when computed individually. However, statistically significant patterns can emerge when the joint associations to multiple related traits are estimated properly. Over the recent years, researchers started recognizing the importance of the joint analysis of multiple correlated phenotypes [5, 18], but there has been a lack of statistical tools to systematically perform such analysis. In our previous work [7], we developed a regularized regression method, called a graph-guided fused lasso, for multitask regression problem that takes advantage of the graph structure over tasks to encourage a selection of common inputs across highly correlated traits in the graph. However, this method can only be applied to the restricted case of correlated continuous-valued outputs. In reality, the set of clinical traits related to a disease often contains both continuous- and discrete-valued traits. As we

demonstrate in our experiments, the $L_1/L_q$ regularization for the joint regression and classification can successfully handle this situation.

The paper is organized as follows. In Section 2, we introduce the notation and the basic formulation for joint regression-classification problem, and describe the $L_1/L_\infty$ and $L_1/L_2$ regularized regressions for heterogeneous multitask learning in this setting. In Section 3, we formulate the parameter estimation as a convex optimization problem, and present an interior-point method for solving it. Section 4 presents experimental results on simulated and asthma datasets. In Section 5, we conclude with a brief discussion of future work.

## 2  Joint Multitask Learning of Linear Regressions and Multinomial Logistic Regressions

Suppose that we have $K$ tasks of learning a predictive model for the output variable, given a common set of $P$ input variables. In our joint regression-classification setting, we assume that the $K$ tasks consist of $K_r$ tasks with continuous-valued outputs and $K_c$ tasks with discrete-valued outputs of an arbitrary number of categories.

For each of the $K_r$ regression problems, we assume a linear relationship between the input vector $X$ of size $P$ and the $k$th output $Y_k$ as follows:

$$Y_k = \beta_{k0}^{(r)} + X\boldsymbol{\beta}_k^{(r)} + \epsilon, \quad k = 1, ..., K_r,$$

where $\boldsymbol{\beta}_k^{(r)} = (\beta_{k1}^{(r)}, \dots, \beta_{kP}^{(r)})'$ represents a vector of $P$ regression coefficients for the $k$th regression task, with the superscript $(r)$ indicating that this is a parameter for regression; $\beta_{k0}^{(r)}$ represents the intercept; and $\epsilon$ denotes the residual.

Let $\mathbf{y}_k = (y_{k1}, \dots, y_{kN})'$ represent the vector of observations for the $k$th output over $N$ samples; and $\mathbf{X}$ represent an $N \times P$ matrix $\mathbf{X} = (\mathbf{x}_1, \dots, \mathbf{x}_N)'$ of the input shared across all of the $K$ tasks, where $\mathbf{x}_i = (x_{i1}, \dots, x_{iP})'$ denotes the $i$th sample. Given these data, we can estimate the $\boldsymbol{\beta}_k^{(r)}$'s by minimizing the sum of squared error:

$$L_r = \sum_{k=1}^{K_r} (\mathbf{y}_k - \mathbf{1}\beta_{k0}^{(r)} - \mathbf{X}\boldsymbol{\beta}_k^{(r)})' \cdot (\mathbf{y}_k - \mathbf{1}\beta_{k0}^{(r)} - \mathbf{X}\boldsymbol{\beta}_k^{(r)}), \tag{1}$$

where $\mathbf{1}$ is an $N$-vector of 1's.

For the tasks with discrete-valued output, we set up a multinomial (i.e., softmax) logistic regression for each of the $K_c$ tasks, assuming that the $k$th task has $M_k$ categories:

$$P(Y_k = m | \mathbf{X} = \mathbf{x}) = \frac{\exp\left(\beta_{k0}^{(c)} + \mathbf{x}\boldsymbol{\beta}_{km}^{(c)}\right)}{1 + \sum_{l=1}^{M_k-1} \exp\left(\beta_{k0}^{(c)} + \mathbf{x}\boldsymbol{\beta}_{kl}^{(c)}\right)}, \quad \text{for } m = 1, \dots, M_k - 1,$$

$$P(Y_k = M_k | \mathbf{X} = \mathbf{x}) = \frac{1}{1 + \sum_{l=1}^{M_k-1} \exp\left(\beta_{k0}^{(c)} + \mathbf{x}\boldsymbol{\beta}_{kl}^{(c)}\right)}, \tag{2}$$

where $\boldsymbol{\beta}_{km}^{(c)} = (\beta_{km1}^{(c)}, \dots, \beta_{kmP}^{(c)})'$, $m = 1, \dots, (M_k - 1)$, is the parameter vector for the $m$th category of the $k$th classification task, and $\beta_{k0}^{(c)}$ is the intercept.

Assuming that the measurements for the $K_c$ output variables are collected for the same set of $N$ samples as in the regression tasks, we expand each output data $y_{ki}$ for the $k$th task of the $i$th sample into a set of $M_k$ binary variables $\boldsymbol{y}_{ki}' = (y_{k1i}, \dots, y_{kM_ki})$, where each $y_{kmi}$, $m = 1, \dots, M_k$, takes value 1 if the $i$th sample for the $k$th classification task belongs to the $m$th category and value 0 otherwise, and thus $\sum_m y_{kmi} = 1$. Using the observations for the output variable in this representation and the shared input data $\mathbf{X}$, one can estimate the parameters $\boldsymbol{\beta}_{km}^{(c)}$'s by minimizing the negative log-likelihood given as below:

$$L_c = -\sum_{i=1}^{N} \sum_{k=1}^{K_c} \left( \sum_{m=1}^{M_k-1} y_{kmi}(\beta_{k0}^{(c)} + \sum_{j=1}^{P} x_{ij}\beta_{kmj}^{(c)}) - \log\left(1 + \sum_{m=1}^{M_k-1} \exp\left(\beta_{k0}^{(c)} + \sum_{j=1}^{P} x_{ij}\beta_{kmj}^{(c)}\right)\right) \right). \tag{3}$$

In this joint regression-classification problem, we form a global objective function by combining the two empirical loss functions in Equations (1) and (3):

$$L = L_r + L_c. \tag{4}$$

This is equivalent to estimating the $\boldsymbol{\beta}_k^{(r)}$'s and $\boldsymbol{\beta}_{km}^{(c)}$'s independently for each of the $K$ tasks, assuming that there are no shared patterns in the way that each of the $K$ output variables is dependent on the input variables. Our goal is to increase the performance of variable selection and prediction power by allowing the sharing of information among the heterogeneous tasks.

## 3   Heterogeneous Multitask Learning with Joint Sparse Feature Selection

In real-world applications, often the covariates lie in a very high-dimensional space with only a small fraction of them being involved in determining the output, and the goal is to recover the sparse structure in the predictive model by selecting the true relevant covariates. For example, in a genetic association mapping, often millions of genetic markers over a population of individuals are examined to find associations with the given phenotype such as clinical traits, disease status, or molecular phenotypes. The challenge in this type of study is to locate the true causal SNPs that influence the phenotype. We consider the case where the related tasks share the same sparsity pattern such that they have a common set of relevant input variables for both the regression and classification tasks and the amount of influence of the relevant input variables on the output may vary across the tasks. We introduce an $L_1/L_q$ regularization to the problem of the heterogeneous multitask learning in Equation (4) as below:

$$L = L_r + L_c + \lambda P_q, \tag{5}$$

where $P_q$ is the group penalty to the sum of linear regression loss and logistic loss, and $\lambda$ is a regularization parameter which determines the sparsity level and could be chosen by cross validation. We consider two extreme cases of the $L_1/L_q$ penalty for group variable selection in our problem which are $L_\infty$ norm and $L_2$ norm across different tasks in one dimension.

$$P_\infty = \left( \sum_{j=1}^{P} \max_{k,m} \left( |\beta_{kj}^{(r)}|, \ |\beta_{kmj}^{(c)}| \right) \right) \ \text{ or } \ P_2 = \left( \sum_{j=1}^{P} |\boldsymbol{\beta}_j^{(r)}, \boldsymbol{\beta}_j^{(c)}|_{L_2} \right), \tag{6}$$

where $\boldsymbol{\beta}_j^{(r)}, \boldsymbol{\beta}_j^{(c)}$ are vector of parameters over all regression and classification tasks, respectively, for the $j$th dimension. Here, the $L_\infty$ and $L_2$ norms over the parameters across different tasks can regulate the joint sparsity among tasks. The $L_1/L_\infty$ and $L_1/L_2$ norms encourage group sparsity in a similar way in that the $\beta_{kj}^{(r)}$'s and $\beta_{kmj}^{(c)}$'s are set to 0 simultaneously for all of the tasks for dimension $j$ if the $L_\infty$ or $L_2$ norm for that dimension is set to be 0. Similarly, if the $L_1$ operator selects a non-zero value for the $L_\infty$ or $L_2$ norm of the $\beta_{kj}^{(r)}$'s and $\beta_{kmj}^{(c)}$'s for the $j$th input, the same input is considered as relevant possibly to all of the tasks, and the $\beta_{kj}^{(r)}$'s and $\beta_{kmj}^{(c)}$'s can have any non-zero values smaller than the maximum or satisfying the $L_2$-norm constraints. The $L_1/L_\infty$ penalty tends to encourage the parameter values to be the same across all tasks for a given input [17], whereas under $L_1/L_2$ penalty the values of the parameters across tasks tend to be more different for a given input than in the $L_1/L_\infty$ penalty.

## 4   Optimization Method

Different methods such as gradient descent, steepest descent, Newton's method and Quasi-Newton method can be used to solve the problem in Equation (5). Although second-order methods have a fast convergence near the global minimum of the convex objective functions, they involve computing a Hessian matrix and inverting it, which can be infeasible in a high-dimensional setting. The coordinate-descent method iteratively updates each element of the parameter vector one at a time, using a closed-form update equation given all of the other elements. However, since it is a first-order method, the speed of convergence becomes slow as the number of tasks and dimension increase. In [8], the truncated Newton's method that uses a preconditionor and solves the linear system instead of inverting the Hessian matrix has been proposed as a fast optimization method for a very large-scale

problem. The linear regression loss and logistic regression loss have different forms. The interior method optimizes their original loss function without any transformation so that it is more intuitive to see how the two heterogeneous tasks affect each other.

In this section, we discuss the case of the $L_1/L_\infty$ penalty since the same optimization method can be easily extended to handle the $L_1/L_2$ penalty. First, we re-write the problem of minimizing Equation (5) with the nondifferentiable $L_1/L_\infty$ penalty as

$$\text{minimize } L_r + L_c + \lambda \sum_{j=1}^{P} u_j$$

$$\text{subject to } \max_{k,m} \left( |\beta_{kj}^{(r)}|, \ |\beta_{kmj}^{(c)}| \right) < u_j, \text{ for } j = 1, \ldots, P, \ k = 1, \ldots, K_r + K_c. \quad (7)$$

Further re-writing the constraints in the above problem, we obtain $2 \cdot P \cdot (K_r + \sum_{k=1}^{K_c}(M_k - 1))$ inequality constraints as follows:

$$-u_j < \beta_{kj}^{(r)} < u_j, \quad \text{for} \quad k = 1, \ldots, K_r, \ j = 1, \ldots, P,$$

$$-u_j < \beta_{kmj}^{(c)} < u_j, \quad \text{for} \quad k = 1, \ldots, K_c, \ j = 1, \ldots, P, \ m = 1, \ldots, M_k - 1.$$

Using the barrier method [3], we re-formulate the objective function in Equation (7) into an unconstrained problem given as

$$L_{\text{Barrier}} = L_r + L_c + \lambda \sum_{j=1}^{P} u_j + \sum_{k=1}^{K_r} \sum_{j=1}^{P} \left( I_-(-\beta_{kj}^{(c)} - u_j) + I_-(\beta_{kj}^{(c)} - u_j) \right)$$

$$+ \sum_{k=1}^{K_c} \sum_{m=1}^{M_k-1} \sum_{j=1}^{P} I_-(-\beta_{kmj}^{(c)} - u_j) + I_-(\beta_{kmj}^{(c)} - u_j),$$

where

$$I_-(x) = \begin{cases} 0 & x \leq 0 \\ \infty & x > 0 \end{cases}.$$

Then, we apply the log barrier function $I_-(f(x)) = -(1/t) \log(-f(x))$, where $t$ is an additional parameter that determines the accuracy of the approximation.

Let $\Theta$ denote the set of parameters $\beta_k^{(r)}$'s and $\beta_{km}^{(c)}$'s. Given a strictly feasible $\Theta$, $t = t^{(0)} > 0$, $\mu > 1$, and tolerance $\epsilon > 0$, we iterate the following steps until convergence.

**Step 1** Compute $\Theta^*(t)$ by minimizing $L_{\text{Barrier}}$, starting at $\Theta$.

**Step 2** Update: $\Theta := \Theta^*(t)$

**Step 3** Stopping criterion: quit if $m/t < \epsilon$ where $m$ is the number of constraint functions.

**Step 4** Increase $t$: $t := t\mu$

In **Step 1**, we use the Newton's method to minimize $L_{\text{Barrier}}$ at $t$. In each iteration, we increase $t$ in **Step 4**, so that we have a more accurate approximation of $I_-(u)$ through $I_-(f(x)) = -(1/t) \log(-f(x))$.

In **Step 1**, we find the direction towards the optimal solution using Newton's method:

$$H \begin{bmatrix} \Delta\boldsymbol{\beta} \\ \Delta\boldsymbol{u} \end{bmatrix} = -\boldsymbol{g},$$

where $\Delta\boldsymbol{\beta}$ and $\Delta\boldsymbol{u}$ are the searching directions of the model parameters and bounding parameters. The $\boldsymbol{g}$ in the above equation is the gradient vector given as $\mathbf{g} = [\boldsymbol{g}^{(r)}, \boldsymbol{g}^{(c)}, \boldsymbol{g}^{(u)}]^T$, where $\boldsymbol{g}^{(r)}$ has $K_r$ components for regression tasks, $\boldsymbol{g}^{(c)}$ has $K_c \times (M_k - 1)$ components for classification tasks, and $H$ is the Hessian matrix given as:

$$H = \begin{bmatrix} R & 0 & D^{(r)} \\ 0 & L & D^{(c)} \\ D^{(r)} & D^{(c)} & F \end{bmatrix},$$

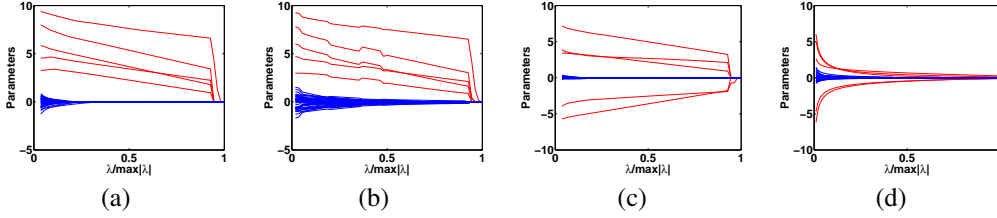

Figure 1: The regularization path for $L_1/L_\infty$-regularized methods. (a) Regression parameters estimated from the heterogeneous task learning method, (b) regression parameters estimated from regression tasks only, (c) logistic-regression parameters estimated from the heterogeneous task learning method, and (d) logistic-regression parameters estimated from classification tasks only. Blue curves: irrelevant inputs; Red curves: relevant inputs.

where $R$ and $L$ are second derivatives of the parameters $\beta$ for regression tasks in the form of $R = \nabla^2 L_r + \nabla^2 P_g|_{\partial\beta^{(r)}\partial\beta^{(r)}}$, $L = \nabla^2 L_c + \nabla^2 P_g|_{\partial\beta^{(c)}\partial\beta^{(c)}}$, $D = \nabla^2 P_g|_{\partial\beta\partial u}$ and $F = D^{(r)} + D^{(c)}$. In the overall interior-point method, the process of constructing and inverting Hessian matrix is the most time-consuming part. In order to make the algorithm scalable to a large problem, we use a preconditionor $\text{diag}(H)$ of the Hessian matrix $H$, and apply the preconditioned conjugate-gradient algorithm to compute the searching direction.

## 5  Experiments

We demonstrate our methods for heterogeneous multitask learning with $L_1/L_\infty$ and $L_1/L_2$ regularizations on simulated and asthma datasets, and compare their performances with those from solving two types of multitask-learning problems for regressions and classifications separately.

### 5.1  Simulation Study

In the context of genetic association analysis, we simulate the input and output data with known model parameters as follows. We start from the 120 haplotypes of chromosome 7 from the population of European ancestry in HapMap data [12], and randomly mate the haplotypes to generate genotype data for 500 individuals. We randomly select 50 SNPs across the chromosome as inputs. In order to simulate the parameters $\beta_k^{(r)}$'s and $\beta_{km}^{(c)}$'s, we assume six regression tasks and a single classification task with five categories, and choose five common SNPs from the total of 50 SNPs as relevant covariates across all of the tasks. We fill the non-zero entries in the regression coefficients $\beta_k^{(r)}$'s with values uniformly distributed in the interval $[a, b]$ with $5 \leq a, b \leq 10$, and the non-zero entries in the logistic-regression parameters $\beta_{km}^{(c)}$'s such that the five categories are separated in the output space. Given these inputs and the model parameters, we generate the output values, using the noise for regression tasks distributed as $N(0, \sigma_{\text{sim}}^2)$. In the classification task, we expand the single output into five dummy variables representing different categories that take values of 0 or 1 depending on which category each sample belongs to. We repeat this whole process of simulating inputs and outputs to obtain 50 datasets, and report the results averaged over these datasets.

The regularization paths of the different multitask-learning methods with an $L_1/L_\infty$ regularization obtained from a single simulated dataset are shown in Figure 1. The results from learning all of the tasks jointly are shown in Figures 1(a) and 1(c) for regression and classification tasks, respectively, whereas the results from learning the two sets of regression and classification tasks separately are shown in Figures 1(b) and 1(d). The red curves indicate the parameters for true relevant inputs, and the blue curves for true irrelevant inputs. We find that when learning both types of tasks jointly, the parameters of the irrelevant inputs are more reliably set to zero along the regularization path than learning the two types of tasks separately.

In order to evaluate the performance of the methods, we use two criteria of sensitivity/specificity plotted as receiver operating characteristic (ROC) curves and prediction errors on test data. To obtain ROC curves, we estimate the parameters, sort the input-output pairs according to the magnitude of the estimated $\beta_{kj}^{(r)}$'s and $\beta_{kmj}^{(c)}$'s, and compare the sorted list with the list of input-output pairs with true non-zero $\beta_{kj}^{(r)}$'s and $\beta_{kmj}^{(c)}$'s.

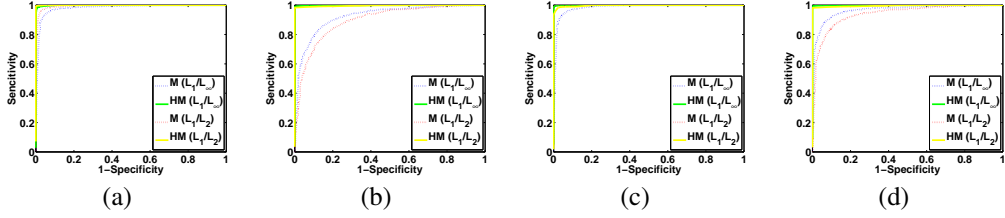

(a)        (b)        (c)        (d)

Figure 2: ROC curves for detecting true relevant input variables when the sample size $N$ varies. (a) Regression tasks with $N = 100$, (b) classification tasks with $N = 100$, (c) regression tasks with $N = 200$, and (d) classification tasks with $N = 200$. Noise level $N(0,1)$ was used. The joint regression-classification methods achieve nearly perfect accuracy, and their ROC curves are completely aligned with the axes. 'M' indicates homogeneous multitask learning, and 'HM' heterogenous multitask learning (This notation is the same for the following other figures).

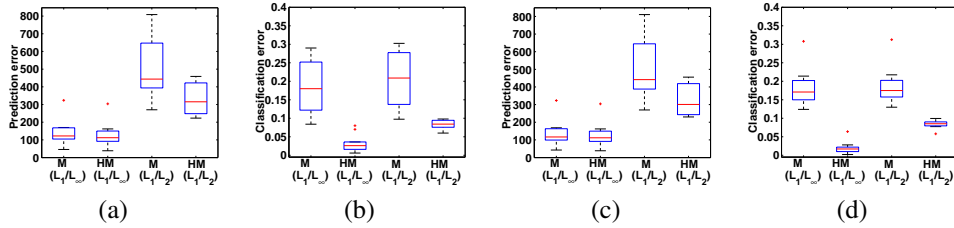

(a)        (b)        (c)        (d)

Figure 3: Prediction errors when the sample size $N$ varies. (a) Regression tasks with $N=100$, (b) classification tasks with $N = 100$, (c) regression tasks with $N = 200$, and (d) classification tasks with $N = 200$. Noise level $N(0,1)$ was used.

We vary the sample size to $N = 100$ and 200, and show the ROC curves for detecting true relevant inputs using different methods in Figure 2. We use $\sigma_{\text{sim}} = 1$ to generate noise in the regression tasks. Results for the regression and classification tasks with $N = 100$ are shown in Figure 2(a) and (b) respectively, and similarly, the results with $N = 200$ in Figure 2(c) and (d). The results with $L_1/L_\infty$ penalty are shown with color blue and green to compare the homogeneous and heterogeneous methods. Red and yellow are results using the $L_1/L_2$ penalty. Although the performance of learning the two types of tasks separately improves with a larger sample size, the joint estimation performs significantly better for both sample sizes. A similar trend can be seen in the prediction errors for the same simulated datasets in Figure 3.

In order to see how different signal-to-noise ratios affect the performance, we vary the noise level to $\sigma^2_{\text{sim}} = 5$ and $\sigma^2_{\text{sim}} = 8$, and plot the ROC curves averaged over 50 datasets with a sample size $N = 300$ in Figure 4. Our results show that for both of the signal-to-noise ratios, learning regression and classification tasks jointly improves the performance significantly. The same observation can be made from the prediction errors in Figure 5. We can see that the $L_1/L_2$ method tends to improve the variable selection, but the tradeoff is that the prediction error will be high when the noise level is low. While $L_1/L_\infty$ has a good balance between the variable selection accuracy and prediction error at a lower noise level, as the noise increases, the $L_1/L_2$ outperforms $L_1/L_\infty$ in both variable selection and prediction accuracy.

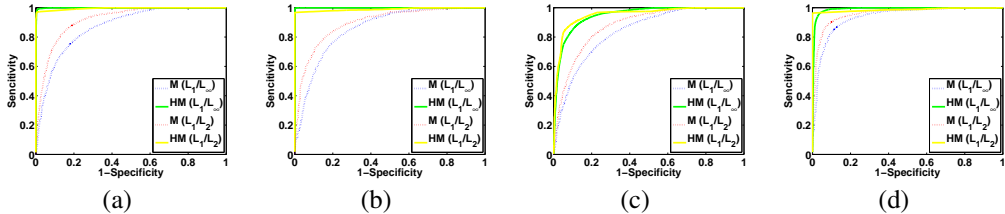

(a)        (b)        (c)        (d)

Figure 4: ROC curves for detecting true relevant input variables when the noise level varies. (a) Regression tasks with noise level $N(0,5)$, (b) classification tasks with noise level $N(0,5)$, (c) regression tasks with noise level $N(0,8)$, and (d) classification tasks with noise level $N(0,8)$. Sample size $N=300$ was used.

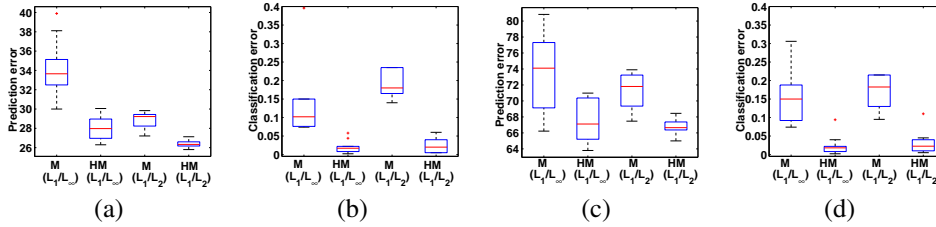

Figure 5: Prediction errors when the noise level varies. (a) Regression tasks with noise level $N(0, 5^2)$, (b) classification tasks with noise level $N(0, 5^2)$, (c) regression tasks with noise level $N(0, 8^2)$, and (d) classification tasks with noise level $N(0, 8^2)$. Sample size $N$=300 was used.

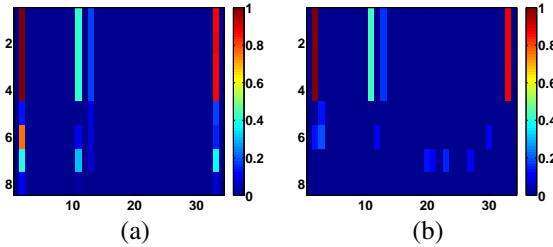

Figure 6: Parameters estimated from the asthma dataset for discovery of causal SNPs for the correlated phenotypes. (a) Heterogeneous task learning method, and (b) separate analysis of multitask regressions and multitask classifications. The rows represent tasks, and the columns represent SNPs.

## 5.2 Analysis of Asthma Dataset

We apply our method to the asthma dataset with 34 SNPs in the *IL4R* gene of chromosome 11 and five asthma-related clinical traits collected over 613 patients. The set of traits includes four continuous-valued traits related to lung physiology such as baseline predrug FEV1, maximum FEV1, baseline predrug FVC, and maximum FVC as well as a single discrete-valued trait with five categories. The goal of this analysis is to discover whether any of the SNPs (inputs) are influencing each of the asthma-related traits (outputs). We fit the joint regression-classification method with $L_1/L_\infty$ and $L_1/L_2$ regularizations, and compare the results from fitting $L_1/L_\infty$ and $L_1/L_2$ regularized methods only for the regression tasks or only for the classification task. We show the estimated parameters for the joint learning with $L_1/L_\infty$ penalty in Figure 6(a) and the separate learning with $L_1/L_\infty$ penalty in Figure 6(b), where the first four rows correspond to the four regression tasks, the next four rows are parameters for the four dummy variables of the classification task, and the columns represent SNPs. We can see that the heterogeneous multitask-learning method encourages to find common causal SNPs for the multiclass classification task and the regression tasks.

## 6 Conclusions

In this paper, we proposed a method for a recovery of union support in heterogeneous multitask learning, where the set of tasks consists of both regressions and classifications. In our experiments with simulated and asthma datasets, we demonstrated that using $L_1/L_2$ or $L_1/L_\infty$ regularizations in the joint regression-classification problem improves the performance for identifying the input variables that are commonly relevant to multiple tasks.

The sparse union support recovery as was presented in this paper is concerned with finding inputs that influence at least one task. In the real-world problem of association mapping, there is a clustering structure such as co-regulated genes, and it would be interesting to discover SNPs that are causal to at least one of the outputs within the subgroup rather than all of the outputs. In addition, SNPs in a region of chromosome are often correlated with each other because of the non-random recombination process during inheritance, and this correlation structure, called linkage disequilibrium, has been actively investigated. A promising future direction would be to model this complex correlation pattern in both the input and output spaces within our framework.

**Acknowledgments** EPX is supported by grant NSF DBI-0640543, NSF DBI-0546594, NSF IIS-0713379, NIH grant 1R01GM087694, and an Alfred P. Sloan Research Fellowship.

# References

[1] A. Argyriou, T. Evgeniou, and M. Pontil. Convex multi-task feature learning. *Machine Learning*, 73(3):243–272, 2008.

[2] B. Bakker and T. Heskes. Task clustering and gating for bayesian multitask learning. *Journal of Machine Learning Research*, 4:83–99, 2003.

[3] S. Boyd and L. Vandenberghe. *Convex Optimization*. Cambridge University Press, 2004.

[4] R. Caruana. Multitask learning. *Machine Learning*, 28:41–75, 1997.

[5] V. Emilsson, G. Thorleifsson, B. Zhang, A.S. Leonardson, F. Zink, J. Zhu, S. Carlson, A. Helgason, G.B. Walters, S. Gunnarsdottir, et al. Variations in dna elucidate molecular networks that cause disease. *Nature*, 452(27):423–28, 2008.

[6] J. Friedman, T. Hastie, and R. Tibshirani. Regularization paths for generalized linear models via coordinate descent. Technical Report 703, Department of Statistics, Stanford University, 2009.

[7] S. Kim and E. P. Xing. Statistical estimation of correlated genome associations to a quantitative trait network. *PLoS Genetics*, 5(8):e1000587, 2009.

[8] K. Koh, S. Kim, and S. Boyd. An interior-point method for large-scale l1-regularized logistic regression. *Journal of Machine Learning Research*, 8(8):1519–1555, 2007.

[9] G. Obozinski, B. Taskar, and M. Jordan. Joint covariate selection for grouped classification. Technical Report 743, Department of Statistics, University of California, Berkeley, 2007.

[10] G. Obozinski, M.J. Wainwright, and M.J. Jordan. High-dimensional union support recovery in multivariate regression. In *Advances in Neural Information Processing Systems 21*, 2008.

[11] M. Schmidt, G. Fung, and R. Rosales. Fast optimization methods for $l_1$ regularization: a comparative study and two new approaches. In *Proceedings of the European Conference on Machine Learning*, 2007.

[12] The International HapMap Consortium. A haplotype map of the human genome. *Nature*, 437:1399–1320, 2005.

[13] R. Tibshirani. Regression shrinkage and selection via the lasso. *Journal of Royal Statistical Society, Series B*, 58(1):267–288, 1996.

[14] K. Yu, V. Tresp, and A. Schwaighofer. Learning gaussian processes from multiple tasks. In *Proceedings of the 22nd International Conference on Machine Learning*, 2005.

[15] M. Yuan and Y. Lin. Model selection and estimation in regression with grouped variables. *Journal of Royal Statistical Society, Series B*, 68(1):49–67, 2006.

[16] J. Zhang, Z. Ghahramani, and Y. Yang. Flexible latent variable models for multi-task learning. *Machine Learning*, 73(3):221–242, 2008.

[17] P. Zhao, G. Rocha, and B. Yu. Grouped and hierarchical model selection through composite absolute penalties. Technical Report 703, Department of Statistics, University of California, Berkeley, 2008.

[18] J. Zhu, B. Zhang, E.N. Smith, B. Drees, R.B. Brem, L. Kruglyak, R.E. Bumgarner, and E.E. Schadt. Integrating large-scale functional genomic data to dissect the complexity of yeast regulatory networks. *Nature Genetics*, 40:854–61, 2008.

